# An Application of Markov Random Fields to Range Sensing

**James Diebel and Sebastian Thrun**

Stanford AI Lab
Stanford University, Stanford, CA 94305

## Abstract

This paper describes a highly successful application of MRFs to the problem of generating high-resolution range images. A new generation of range sensors combines the capture of low-resolution range images with the acquisition of registered high-resolution camera images. The MRF in this paper exploits the fact that discontinuities in range and coloring tend to co-align. This enables it to generate high-resolution, low-noise range images by integrating regular camera images into the range data. We show that by using such an MRF, we can substantially improve over existing range imaging technology.

## 1 Introduction

In recent years, there has been an enormous interest in developing technologies for measuring range. The set of commercially available technologies include passive stereo with two or more cameras, active stereo, triangulating light stripers, millimeter wavelength radar, and scanning and flash lidar. In the low-cost arena, systems such as the *Swiss Ranger* and the *CanestaVision* sensors provide means to acquire low-res range data along with passive camera images. Both of these devices capture high-res visual images along with lower-res depth information. This is the case for a number of devices at all price ranges, including the highly-praised range camera by *3DV Systems*.

This paper addresses a single shortcoming that (with the exception of stereo) is shared by most active range acquisition devices: Namely that range is captured at much lower resolution than images. This raises the question as to whether we *can turn a low-resolution depth imager into a high-resolution one, by exploiting conventional camera images?* A positive answer to this question would significantly advance the field of depth perception. Yet we lack techniques to fuse high-res conventional images with low-res depth images.

This paper applies graphical models to the problem of fusing low-res depth images with high-res camera images. Specifically, we propose a Markov Random Field (MRF) method for integrating both data sources. The intuition behind the MRF is that depth discontinuities in a scene often co-occur with color or brightness changes within the associated camera image. Since the camera image is commonly available at much higher resolution, this insight can be used to enhance the resolution and accuracy of the depth image.

Our approach performs this data integration using a multi-resolution MRF, which ties together image and range data. The mode of the probability distribution defined by the MRF provides us with a high-res depth map. Because we are only interested in finding the mode, we can apply fast optimization technique to the MRF inference problem, such as a

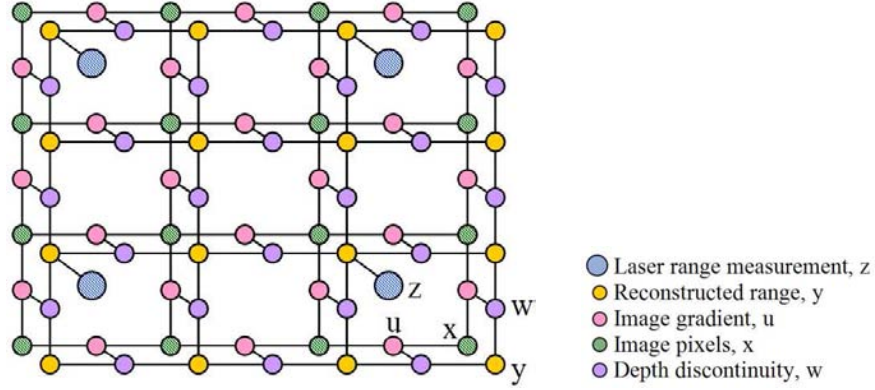

**Figure 1**: The MRF is composed of 5 node types: The measurements mapped to two types of variables, the range measurement variables labeled $z$, image pixel variables labeled $x$. The density of image pixels is larger than those of the range measurements. The reconstructed range nodes, labeled $y$, are unobservable, but their density matches that of the image pixels. Auxiliary nodes labeled $w$ and $u$ mediate the information from the image and the depth map, as described in the text.

conjugate gradient algorithm. This approach leads to a high-res depth map within seconds, increasing the resolution of our depth sensor by an order of magnitude while improving local accuracy. To back up this claim, we provide several example results obtained using a low-res laser range finder paired with a conventional point-and-shot camera.

While none of the modeling or inference techniques in this paper are new, we believe that this paper provides a significant application of graphical modeling techniques to a problem that can dramatically alter an entire growing industry.

## 2   The Image-Range MRF

Figure 1 shows the MRF designed for our task. The input to the MRF occurs at two layers, through the variables labeled $x_i$ and the variables labeled $z_i$. The variables $x_i$ correspond to the image pixels, and their values are the three-dimensional RGB value of each pixel. The variables $z_i$ are the range measurements. The range measurements are sampled much less densely than the image pixels, as indicated in this figure.

The key variables in this MRF are the ones labeled $y$, which model the reconstructed range at the same resolution as the image pixels. These variables are unobservable. Additional nodes labeled $u$ and $w$ leverage the image information into the estimated depth map $y$.

Specifically, the MRF is defined through the following potentials:

1. The *depth measurement potential* is of the form

$$\Psi \quad = \quad \sum_{i \in L} k \, (y_i - z_i)^2 \tag{1}$$

   Here $L$ is the set of indexes for which a depth measurement is available, and $k$ is a constant weight placed on the depth measurements. This potential measures the quadratic distance between the estimated range in the high-res grid $y$ and the measured range in the variables $z$, where available.

2. A *depth smoothness prior* is expressed by a potential of the form

$$\Phi \quad = \quad \sum_{i} \sum_{j \in N(i)} w_{ij} \, (y_i - y_j)^2 \tag{2}$$

Here $N(i)$ is the set of nodes adjacent to $i$. $\Phi$ is a weighted quadratic distance between neighboring nodes.

3. The *weighting factors* $w_{ij}$ are a key element, in that they provide the link to the image layer in the MRF. Each $w_{ij}$ is a deterministic function of the corresponding two adjacent image pixels, which is calculated as follows:

$$w_{ij} \quad = \quad \exp(-c\,u_{ij}) \tag{3}$$

$$u_{ij} \quad = \quad ||x_i - x_j||_2^2 \tag{4}$$

Here $c$ is a constant that quantifies how unwilling we are to have smoothing occur across edges in the image.

The resulting MRF is now defined through the constraints $\Psi$ and $\Phi$. The conditional distribution over the target variables $y$ is given by an expression of the form

$$p(y \mid x, z) \quad = \quad \frac{1}{Z}\, \exp(-\frac{1}{2}(\Psi + \Phi)) \tag{5}$$

where $Z$ is a normalizer (partition function).

## 3   Optimization

Unfortunately, computing the full posterior is impossible for such an MRF, not least because the MRF may possesses many millions of nodes; even loopy belief propagation [19] requires enormous time for convergence. Instead, for the depth reconstruction problem we shall be content with computing the *mode* of the posterior.

Finding the mode of the log-posterior is, of course, a least square optimization problem, which we solve with the well-known conjugate gradient (CG) algorithm [12]. A typical single-image optimization with $2 \cdot 10^5$ nodes takes about a second to optimize on a modern computer.

The details of the CG algorithm are omitted for brevity, but can be found in contemporary texts. The resulting algorithm for probable depth image reconstruction is now remarkably simple: Simply set $y^{[0]}$ by the bilinear interpolation of $z$, and then iterate the CG update rule. The result is a probable reconstruction of the depth map at the same resolution as the camera image.

## 4   Results

Our experiments were performed with a SICK sweeping laser range finder and a Canon consumer digital camera with 5 mega pixels per image. Both were mounted on a rotating platform controlled by a servo actuator. This configuration allows us to survey an entire room from a consistent vantage point and with known camera and laser positions at all times. The output of this system is a set of pre-aligned laser range measurements and camera images.

Figure 2 shows a scan of a bookshelf in our lab. The top row contains several views of the raw measurements and the bottom row is the output of the MRF. The latter is clearly much sharper and less noisy; many features that are smaller than the resolution of the laser scanner are pulled out by the camera image. Figure 5 shows the same scene from much further back.

A more detailed look is taken in Figure 3. Here we examine the painted metal door frame to an office. The detailed structure is completely invisible in the raw laser scan but is easily drawn out when the image data is incorporated. It is notable that traditional mesh fairing algorithms would not be able to recover this fine structure, as there is simply insufficient evidence of it in the range data alone. Specifically, when running our MRF using a *fixed* value for $w_{ij}$, which effectively decouples the range image and the depth image, the depth reconstruction leads to a model that is either overly noise (for $w_{ij} = 1$ or

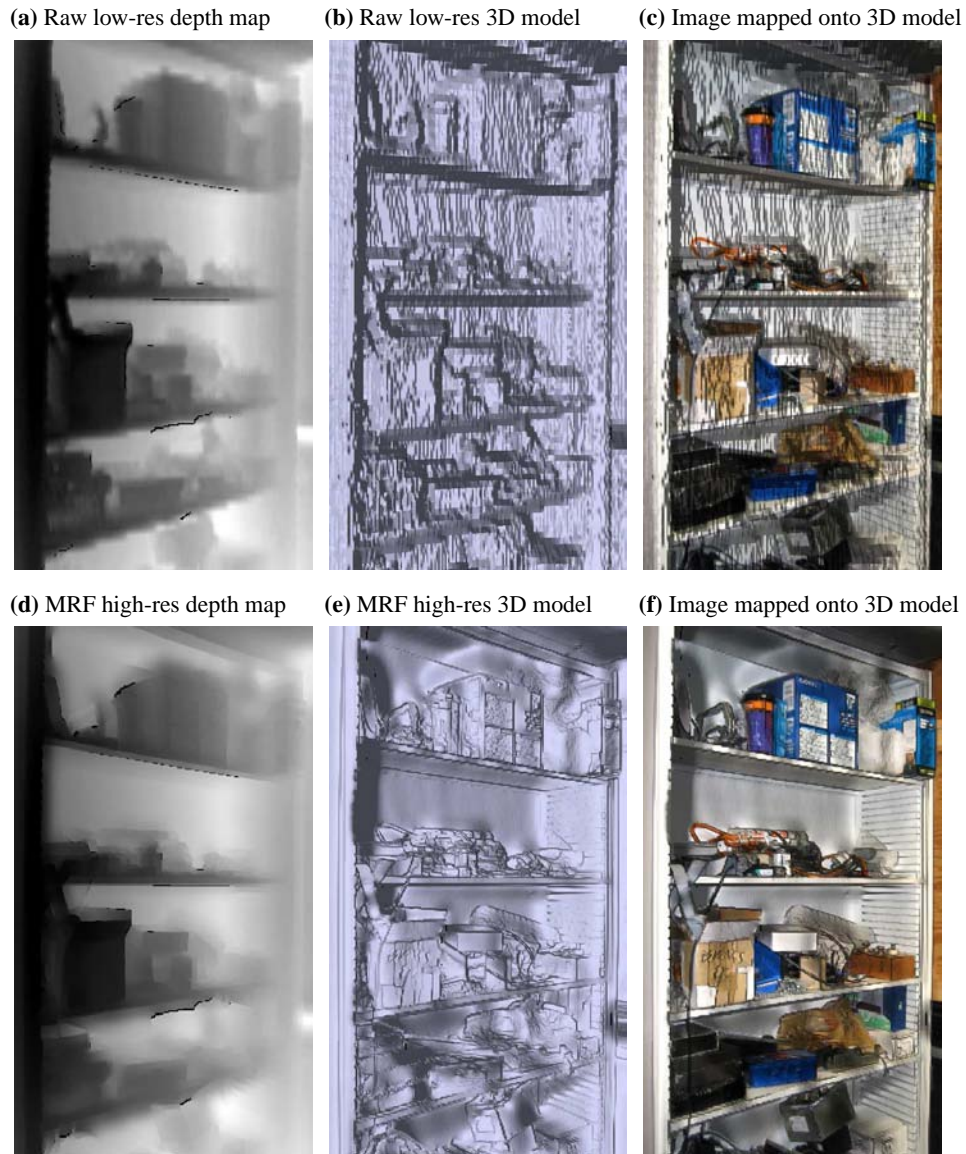

**(a)** Raw low-res depth map    **(b)** Raw low-res 3D model    **(c)** Image mapped onto 3D model

**(d)** MRF high-res depth map    **(e)** MRF high-res 3D model    **(f)** Image mapped onto 3D model

**Figure 2**: Example result of our MRF approach. Panels (a-c) show the raw data, the low-res depth map, a 3D model constructed from this depth map, and the same model with image texture superimposed. Panels (d-f) show the results of our algorithm. The depth map is now high-resolution, as is the 3D model. The 3D rendering is a substantial improvement over the raw sensor data; in fact, many small details are now visible.

smooths out the edge features for $w_{ij} = 5$. Our approach clearly recovers those corners, thanks to the use of the camera image.

Finally, in Fig. 4 we give one more example of a shipping crate next to a white wall. The coarse texture of the wooden surface is correctly inferred in contrast to the smooth white wall. This brings up the obvious problem that sharp color gradients do frequently occur on smooth surfaces; take, for example, posters. While this fact can sometimes lead to falsely-textured surfaces, it has been our experience that these flaws are often unnoticeable

**(a)** Raw 3D model, with and without color from the image

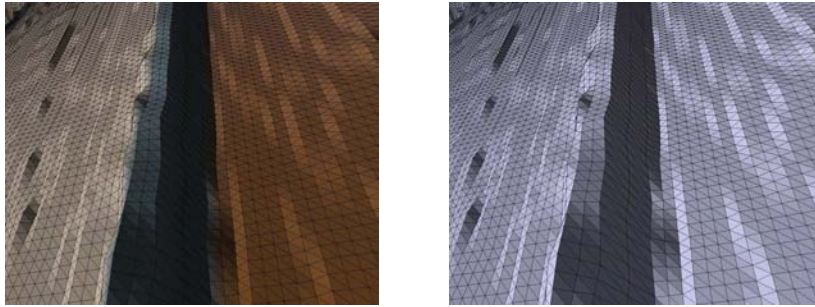

**(b)** Two results ignoring the image color information, for two different smoothers

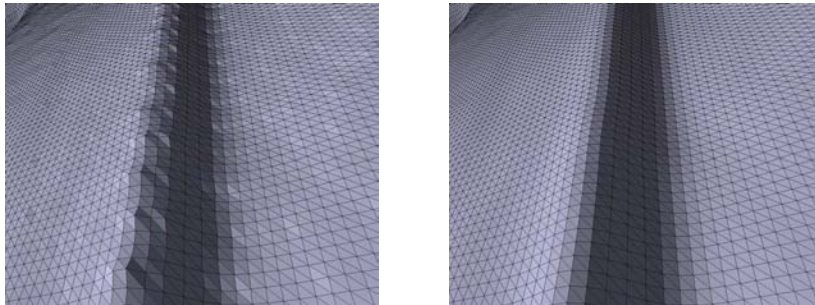

**(c)** Reconstruction with our MRF, integrating both depth and image color

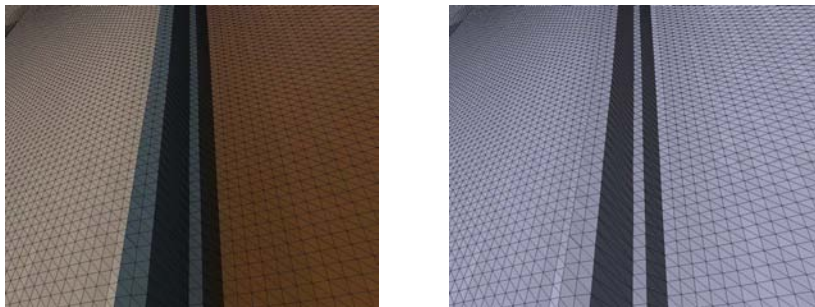

**Figure 3**: The important of the image information in depth recovery is illustrated in this figure. It shows a part of a door frame, for which a course depth map and a fine-grained image is available. The rendering labeled (b) show the result of our MRF when color is entirely ignored, for different fixed value of the weights $w_{ij}$. The images in (c) are the results of our approach, which clearly retains the sharp corner of the door frame.

and certainly no worse than the original scan. Clearly, the reconstruction of such depth maps is an ill-posed problem, and our approach generates a high-res model that is still significantly better than the original data. Notice, however, that the background wall is recovered accurately, and the corner of the room is visually enhanced.

## 5 Related Work

One of the primary acquisition techniques for depth is stereo. A good survey and comparison of stereo algorithms can is due to [14]. Our algorithm does not apply to stereo vision, since by definition the resolution of the image and the inferred depth map are equivalent.

**(a)** 3D model based on the raw range data, with and without texture

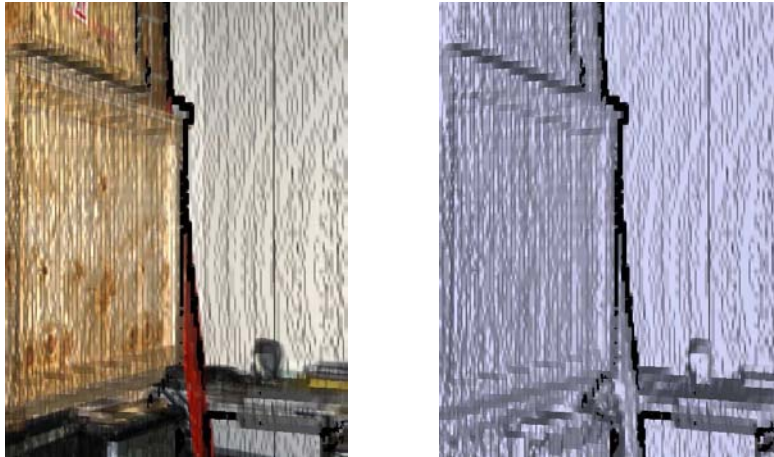

**(b)** Refined and super-resolved model, generated by our MRF

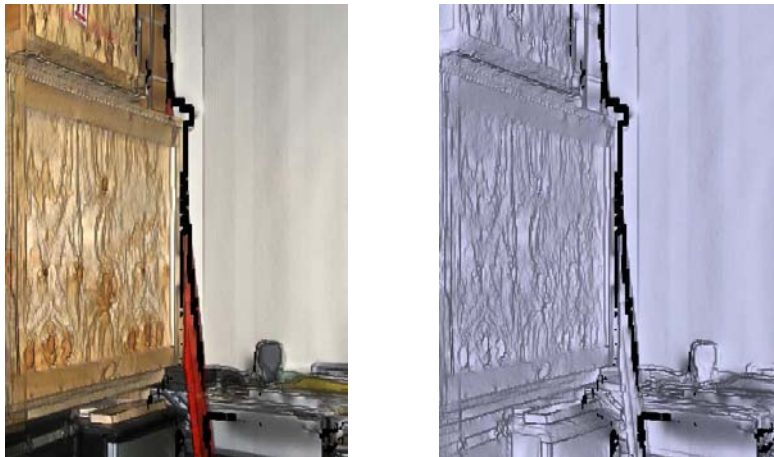

**Figure 4**: This example illustrate that the amount of smoothing in the range data is dependent on the image texture. On the left is a wooden box with an unsmooth surface that causes significant color variations. The 3D model generated from the MRF provides relatively little smoothing. In the background is a while wall with almost no color variation. Here our approach smooths the mesh significantly; in fact, it enhances the visibility of the room corner.

Passive stereo, in which the sensor does not carry its own light source, is unable to estimate ranges in the absence of texture (e.g., when imaging a featureless wall). Active stereo techniques supply their own light [4]. However, those techniques differ in characteristics from laser-based system to an extent that renders them practically inapplicable for many applications (most notably: long-range acquisition, where time-of-flight techniques are an order of magnitude more accurate then triangulation techniques, and bright-light outdoor environments). We remark that Markov Random fields have become a defining methodology in stereo reconstruction [15], along with layered EM-style methods [2, 16]; see the comparison in [14].

Similar work due to [20] relies on a different set of image cues to improve stereo shape estimates. In particular, learned regression coefficients are used to predict the band-passed shape of a scene from a band-passed image of that scene. The regression coefficients are

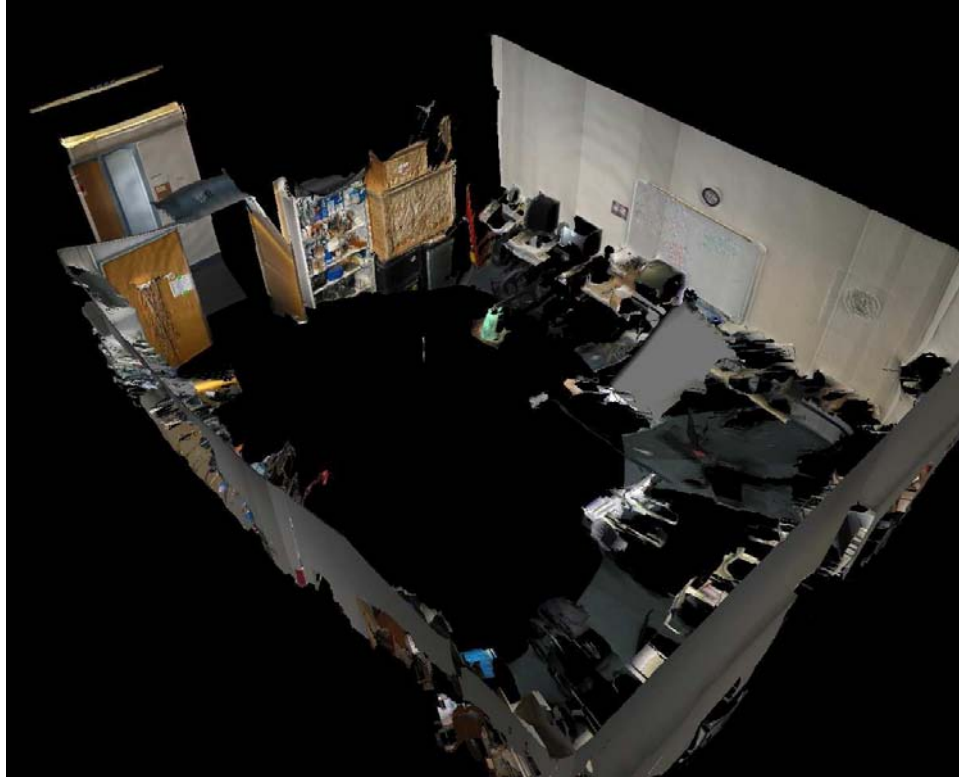

**Figure 5**: 3D model of a larger indoor environment, after applying our MRF.

learned from laser-stripe-scanned reference models with regitered images.

For range images, surfaces, and point clouds, there exists a large literature on smoothing while preserving features such as edges. This includes work on diffusion processes [6], frequency-domain filtering [17], and anisotropic diffusion [5]; see also [3] and [1]. Most recently [10] proposed an efficient non-iterative technique for feature-preserving mesh smoothing, [9] adapted bilateral filtering for application to mesh denoising. and [7] developed anisotropic MRF techniques. None of these techniques, however, integrates high-resolution images to guide the smoothing process. Instead, they all operate on monochromatic 3D surfaces.

Our work can be viewed as generating super-resolution. Super-resolution techniques have long been popular in the computer vision field [8] and in aerial photogrammetry [11]. Here Bayesian techniques are often brought to bear for integrating multiple images into a single image of higher resolution. None of these techniques deal with range data. Finally, multiple range scans are often integrated into a single model [13, 18], yet none of these techniques involve image data.

## 6 Conclusion

We have presented a Markov Random Field that integrated high-res image data into low-res range data, to recover range data at the same resolution as the image data. This approach is specifically aimed at a new wave of commercially available sensors, which provide range at lower resolution than image data.

The significance of this work lies in the results. We have shown that our approach can truly fill the resolution gap between range and images, and use image data to effectively

boost the resolution of a range finder. While none of the techniques used here are new (even though CG is usually not applied for inference in MRFs), we believe this is the first application of MRF to multimodal data integration. A large number of scientific fields would benefit from better range sensing; the present approach provides a solution that endows low-cost range finders with unprecedented resolution and accuracy.

## References

[1] C.L. Bajaj and G. Xu. Anisotropic diffusion of surfaces and functions on surfaces. In *Proceedings of SIGGRAPH*, pages 4–32, 2003.

[2] S. Baker, R Szeliski, and P. Anandan. A layered approach to stereo reconstruction. In *Proceedings of the Conference on Computer Vision and Pattern Recognition (CVPR)*, pages 434–438, Santa Barbara, CA, 1998.

[3] U. Clarenz, U. Diewald, and M. Rumpf. Anisotropic geometric diffusion in surface processing. In *Proceedings of the IEEE Conference on Visualization*, pages 397–405, 2000.

[4] J. Davis, R. Ramamoothi, and S. Rusinkiewicz. Spacetime stereo: A unifying framework for depth from triangulation. In *Proceedings of the Conference on Computer Vision and Pattern Recognition (CVPR)*, 2003.

[5] M. Desbrun, M. Meyer, P. Schröder, and A. Barr. Anisotropic feature-preserving denoising of height fields and bivariate data. In *Proceedings Graphics Interface*, Montreal, Quebec, 2000.

[6] M. Desbrun, M. Meyer, P. Schröder, and A. H. Barr. Implicit fairing of irregular meshes using diffusion and curvature flow. In *Proceedings of SIGGRAPH*, 1999.

[7] J. Diebel, S. Thrun, and M. Brüning. A bayesian method for probable surface reconstruction and decimation. *IEEE Transactions on Graphics*, 2005. To appear.

[8] M. Elad and A. Feuer. Restoration of single super-resolution image from several blurred. *IEEE Transcation on Image Processing*, 6(12):1646–1658, 1997.

[9] S. Fleishman, I. Drori, and D. Cohen-Or. Bilateral mesh denoising. In *Proceedings of SIGGRAPH*, pages 950–953, 2003.

[10] T.R. Jones, F. Durand, and M. Desbrun. Non-iterative, feature-preserving mesh smoothing. In *Proceedings of SIGGRAPH*, pages 943–949, 2003.

[11] I. K. Jung and S. Lacroix. High resolution terrain mapping using low altitude aerial stereo imagery. In *Proceedings of the International Conference on Computer Vision (ICCV)*, Nice, France, 2003.

[12] W. H. Press. *Numerical recipes in C: the art of scientific computing*. Cambridge University Press, Cambridge; New York, 1988.

[13] S. Rusinkiewicz and M. Levoy. Efficient variants of the ICP algorithm. In *Proc. Third International Conference on 3D Digital Imaging and Modeling (3DIM)*, Quebec City, Canada, 2001. IEEEComputer Society.

[14] D. Scharstein and R. Szeliski. A taxonomy and evaluation of dense two-frame stereo correspondence algorithms. *International Journal of Computer Vision*, 47(1-3):7–42, 2002.

[15] J. Sun, H.-Y. Shum, and N.-N. Zheng. Stereo matching using belief propagation. *IEEE Transcation on PAMI*, 25(7), 2003.

[16] R. Szeliski. Stereo algorithms and representations for image-based rendering. In *Proceedings of the British Machine Vision Conference, Vol 2*, pages 314–328, 1999.

[17] G. Taubin. A signal processing approach to fair surface design. In *Proceedings of SIGGRAPH*, pages 351–358, 1995.

[18] S. Thrun, W. Burgard, and D. Fox. *Probabilistic Robotics*. MIT Press, Cambridge, MA, 2005.

[19] Y. Weiss and W.T. Freeman. Correctness of belief propagation in gaussian graphical models of arbitrary topology. *Neural Computation*, 13(10):2173–2200, 2001.

[20] W. T. Freeman and A. Torralba. Shape recipes: Scene representations that refer to the image. In *Advances in Neural Information Processing Systems (NIPS) 15*, Cambridge, MA, 2003. MIT Press.
